# Rational Kernels

**Corinna Cortes    Patrick Haffner    Mehryar Mohri**
AT&T Labs – Research
180 Park Avenue, Florham Park, NJ 07932, USA
{corinna, haffner, mohri}@research.att.com

## Abstract

We introduce a general family of kernels based on weighted transducers or rational relations, *rational kernels*, that can be used for analysis of variable-length sequences or more generally weighted automata, in applications such as computational biology or speech recognition. We show that rational kernels can be computed efficiently using a general algorithm of composition of weighted transducers and a general single-source shortest-distance algorithm. We also describe several general families of positive definite symmetric rational kernels. These general kernels can be combined with Support Vector Machines to form efficient and powerful techniques for spoken-dialog classification: highly complex kernels become easy to design and implement and lead to substantial improvements in the classification accuracy. We also show that the string kernels considered in applications to computational biology are all specific instances of rational kernels.

## 1   Introduction

In many applications such as speech recognition and computational biology, the objects to study and classify are not just fixed-length vectors, but variable-length sequences, or even large sets of alternative sequences and their probabilities. Consider for example the problem that originally motivated the present work, that of classifying speech recognition outputs in a large spoken-dialog application. For a given speech utterance, the output of a large-vocabulary speech recognition system is a weighted automaton called a *word lattice* compactly representing the possible sentences and their respective probabilities based on the models used. Such lattices, while containing sometimes just a few thousand transitions, may contain hundreds of millions of paths each labeled with a distinct sentence.

The application of discriminant classification algorithms to word lattices, or more generally weighted automata, raises two issues: that of handling variable-length sequences, and that of applying a classifier to a distribution of alternative sequences. We describe a general technique that solves both of these problems.

Kernel methods are widely used in statistical learning techniques such as Support Vector Machines (SVMs) [18] due to their computational efficiency in high-dimensional feature spaces. This motivates the introduction and study of kernels for weighted automata. We present a general family of kernels based on weighted transducers or rational relations, *rational kernels* which apply to weighted automata. We show that rational kernels can be computed efficiently using a general algorithm of composition of weighted transducers and a general single-source shortest-distance algorithm.

We also briefly describe some specific rational kernels and their applications to spoken-dialog classification. These kernels are symmetric and positive definite and can thus be combined with SVMs to form efficient and powerful classifiers. An important benefit of

| SEMIRING | SET | $\oplus$ | $\otimes$ | $\overline{0}$ | $\overline{1}$ |
|---|---|---|---|---|---|
| Boolean | $\{0,1\}$ | $\vee$ | $\wedge$ | 0 | 1 |
| Probability | $\mathbb{R}_+$ | $+$ | $\times$ | 0 | 1 |
| Log | $\mathbb{R} \cup \{-\infty, +\infty\}$ | $\oplus_{\log}$ | $+$ | $+\infty$ | 0 |
| Tropical | $\mathbb{R} \cup \{-\infty, +\infty\}$ | min | $+$ | $+\infty$ | 0 |

Table 1: *Semiring examples.* $\oplus_{\log}$ *is defined by:* $x \oplus_{\log} y = -\log(e^{-x} + e^{-y})$.

our approach is its generality and its simplicity: the same efficient algorithm can be used to compute arbitrarily complex rational kernels. This makes highly complex kernels easy to use and helps us achieve substantial improvements in classification accuracy.

## 2 Weighted automata and transducers

In this section, we present the algebraic definitions and notation necessary to introduce rational kernels.

**Definition 1 ([7])** *A system* $(\mathbb{K}, \oplus, \otimes, \overline{0}, \overline{1})$ *is a* semiring *if:* $(\mathbb{K}, \oplus, \overline{0})$ *is a commutative monoid with identity element* $\overline{0}$*;* $(\mathbb{K}, \otimes, \overline{1})$ *is a monoid with identity element* $\overline{1}$*;* $\otimes$ *distributes over* $\oplus$*; and* $\overline{0}$ *is an annihilator for* $\otimes$*: for all* $a \in \mathbb{K}$, $a \otimes \overline{0} = \overline{0} \otimes a = \overline{0}$.

Thus, a semiring is a ring that may lack negation. Table 2 lists some familiar examples of semirings. In addition to the Boolean semiring and the probability semiring used to combine probabilities, two semirings often used in applications are the *log semiring* which is isomorphic to the probability semiring via a log morphism, and the *tropical semiring* which is derived from the log semiring using the Viterbi approximation.

**Definition 2** *A* weighted finite-state transducer $T$ *over a semiring* $\mathbb{K}$ *is an 8-tuple* $T = (\Sigma, \Delta, Q, I, F, E, \lambda, \rho)$ *where:* $\Sigma$ *is the finite input alphabet of the transducer;* $\Delta$ *is the finite output alphabet;* $Q$ *is a finite set of states;* $I \subseteq Q$ *the set of initial states;* $F \subseteq Q$ *the set of final states;* $E \subseteq Q \times (\Sigma \cup \{\epsilon\}) \times (\Delta \cup \{\epsilon\}) \times \mathbb{K} \times Q$ *a finite set of transitions;* $\lambda : I \to \mathbb{K}$ *the initial weight function; and* $\rho : F \to \mathbb{K}$ *the final weight function mapping* $F$ *to* $\mathbb{K}$.

*Weighted automata* can be formally defined in a similar way by simply omitting the input or the output labels.

Given a transition $e \in E$, we denote by $p[e]$ its origin or previous state and $n[e]$ its destination state or next state, and $w[e]$ its weight. A *path* $\pi = e_1 \cdots e_k$ is an element of $E^*$ with consecutive transitions: $n[e_{i-1}] = p[e_i]$, $i = 2, \ldots, k$. We extend $n$ and $p$ to paths by setting: $n[\pi] = n[e_k]$ and $p[\pi] = p[e_1]$. The weight function $w$ can also be extended to paths by defining the weight of a path as the $\otimes$-product of the weights of its constituent transitions: $w[\pi] = w[e_1] \otimes \cdots \otimes w[e_k]$. We denote by $P(q, q')$ the set of paths from $q$ to $q'$ and by $P(q, x, y, q')$ the set of paths from $q$ to $q'$ with input label $x \in \Sigma^*$ and output label $y$ (transducer case). These definitions can be extended to subsets $R, R' \subseteq Q$, by: $P(R, x, y, R') = \cup_{q \in R, q' \in R'} P(q, x, y, q')$.

A transducer $T$ is *regulated* if the output weight associated by $T$ to any pair of input-output string $(x, y)$ by:

$$\llbracket T \rrbracket(x, y) = \bigoplus_{\pi \in P(I, x, y, F)} \lambda(p[\pi]) \otimes w[\pi] \otimes \rho[n[\pi]] \tag{1}$$

is well-defined and in $\mathbb{K}$. $\llbracket T \rrbracket(x) = \overline{0}$ when $P(I, x, y, F) = \emptyset$. In the following, we will assume that all the transducers considered are regulated. Weighted transducers are closed under $\oplus$, $\otimes$ and Kleene-closure. In particular, the $\oplus$-sum and $\otimes$-multiplications of two transducers $T_1$ and $T_2$ are defined for each pair $(x, y)$ by:

$$\llbracket T_1 \oplus T_2 \rrbracket(x, y) = \llbracket T_1 \rrbracket(x, y) \oplus \llbracket T_2 \rrbracket(x, y) \tag{2}$$

$$\llbracket T_1 \otimes T_2 \rrbracket(x, y) = \bigoplus_{x = x_1 x_2, y = y_1 y_2} \llbracket T_1 \rrbracket(x_1, y_1) \otimes \llbracket T_2 \rrbracket(x_2, y_2) \tag{3}$$

# 3 Rational kernels

This section introduces *rational kernels*, presents a general algorithm for computing them efficiently and describes several examples of rational kernels.

## 3.1 Definition

**Definition 3** *A kernel $K$ is* rational *if there exist a weighted transducer $T = (\Sigma, \Delta, Q, I, F, E, \lambda, \rho)$ over the semiring $\mathbb{K}$ and a function $\psi : \mathbb{K} \to \mathbb{R}$ such that for all $x, y \in \Sigma^*$:*

$$K(x, y) = \psi([\![T]\!](x, y)) \tag{4}$$

In general, $\psi$ is an arbitrary function mapping $\mathbb{K}$ to $\mathbb{R}$. In some cases, it may be desirable to assume that it is a semiring morphism as in Section 3.6. It is often the identity function when $\mathbb{K} = \mathbb{R}$ and may be a projection when the semiring $\mathbb{K}$ is the cross-product of $\mathbb{R}$ and another semiring ($\mathbb{K} = \mathbb{R} \times \mathbb{K}'$).

Rational kernels can be naturally extended to kernels over weighted automata. In the following, to simplify the presentation, we will restrict ourselves to the case of acyclic weighted automata which is the case of interest for our applications, but our results apply similarly to arbitrary weighted automata. Let $A$ and $B$ be two acyclic weighted automata over the semiring $\mathbb{K}$, then $K(A, B)$ is defined by:

$$K(A, B) = \psi(\bigoplus_{x, y} [\![A]\!](x) \otimes [\![T]\!](x, y) \otimes [\![B]\!](y)) \tag{5}$$

More generally, the results mentioned in the following for strings apply all similarly to acyclic weighted automata. Since the set of weighted transducers over a semiring $\mathbb{K}$ is also closed under $\oplus$-sum and $\otimes$-product [2, 3], it follows that rational kernels over a semiring $\mathbb{K}$ are closed under sum and product. We denote by $K_1 + K_2$ the sum and by $K_1 \times K_2$ the product of two rational kernels $K_1$ and $K_2$. Let $T_1$ and $T_2$ be the associated transducers of these kernels, we have for example:

$$(K_1 + K_2)(x, y) = \psi([\![(T_1 \oplus T_2)]\!](x, y)) = K_1(x, y) + K_2(x, y) \tag{6}$$

In learning techniques such as those based on SVMs, we are particularly interested in positive definite symmetric kernels, which guarantee the existence of a corresponding reproducing kernel Hilbert space. Not all rational kernels are positive definite symmetric but in the following sections we will describe some general classes of rational kernels that have this property.

Positive definite symmetric kernels can be used to construct other families of kernels that also meet these conditions [17]. *Polynomial kernels* of degree $p$ are formed from the expression $(K + a)^p$, and *Gaussian kernels* can be formed as $\exp(-d^2/\sigma^2)$ with $d^2(x, y) = K(x, x) + K(y, y) - 2K(x, y)$. Since the class of symmetric positive definite kernels is closed under sum [1], the sum of two positive definite rational kernels is also a positive definite rational kernel.

In what follows, we will focus on the algorithm for computing rational kernels. The algorithm for computing $K(x, y)$, or $K(A, B)$, for any two acyclic weighted automata, is based on two general algorithms that we briefly present: composition of weighted transducers to combine $A$, $T$, and $B$, and a general shortest-distance algorithm in a semiring $\mathbb{K}$ to compute the $\oplus$-sum of the weights of the successful paths of the combined machine.

## 3.2 Composition of weighted transducers

Composition is a fundamental operation on weighted transducers that can be used in many applications to create complex weighted transducers from simpler ones. Let $\mathbb{K}$ be a commutative semiring and let $T_1$ and $T_2$ be two weighted transducers defined over $\mathbb{K}$ such that the input alphabet of $T_2$ coincides with the output alphabet of $T_1$. Then, the composition of $T_1$ and $T_2$ is a weighted transducer $T_1 \circ T_2$ which, when it is regulated, is defined for all

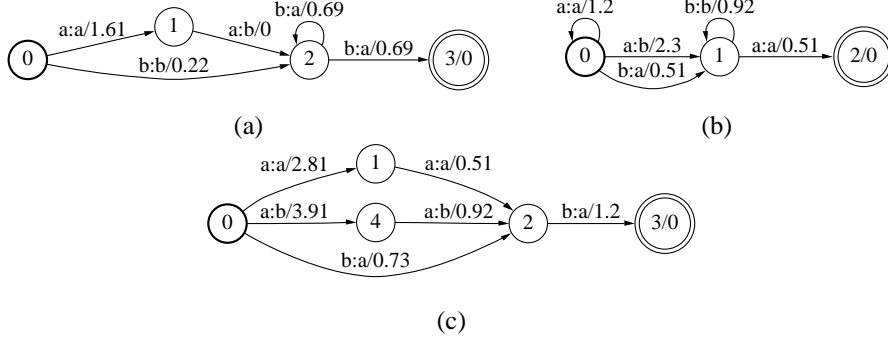

Figure 1: *(a) Weighted transducer $T_1$ over the log semiring. (b) Weighted transducer $T_2$ over the log semiring. (c) Construction of the result of composition $T_1 \circ T_2$. Initial states are represented by bold circles, final states by double circles. Inside each circle, the first number indicates the state number, the second, at final states only, the value of the final weight function $\rho$ at that state. Arrows represent transitions and are labeled with symbols followed by their corresponding weight.*

$x, y$ by [2, 3, 15, 7]:[1]

$$[\![T_1 \circ T_2]\!](x, y) = \bigoplus_z [\![T_1]\!](x, z) \otimes [\![T_2]\!](z, y) \tag{7}$$

Note that a transducer can be viewed as a matrix over a countable set $\Sigma^* \times \Delta^*$ and composition as the corresponding matrix-multiplication. There exists a general and efficient composition algorithm for weighted transducers which takes advantage of the sparsity of the input transducers [14, 12]. States in the composition $T_1 \circ T_2$ of two weighted transducers $T_1$ and $T_2$ are identified with pairs of a state of $T_1$ and a state of $T_2$. Leaving aside transitions with $\epsilon$ inputs or outputs, the following rule specifies how to compute a transition of $T_1 \circ T_2$ from appropriate transitions of $T_1$ and $T_2$:[2]

$$(q_1, a, b, w_1, q_2) \quad \text{and} \quad (q'_1, b, c, w_2, q'_2) \Longrightarrow ((q_1, q'_1), a, c, w_1 \otimes w_2, (q_2, q'_2)) \tag{8}$$

In the worst case, all transitions of $T_1$ leaving a state $q_1$ match all those of $T_2$ leaving state $q'_1$, thus the space and time complexity of composition is quadratic: $O((|Q_1|+|E_1|)(|Q_2|+|E_2|))$. Fig.(1) (a)-(c) illustrate the algorithm when applied to the transducers of Fig.(1) (a)-(b) defined over the log semiring. The *intersection* of two weighted automata is a special case of composition. It corresponds to the case where the input and output label of each transition are identical.

### 3.3 Single-source shortest distance algorithm over a semiring

Given a weighted automaton or transducer $M$, the *shortest-distance* from state $q$ to the set of final states $F$ is defined as the $\oplus$-sum of all the paths from $q$ to $F$:

$$d[q] = \bigoplus_{\pi \in P(q, F)} w[\pi] \otimes \rho[n[\pi]] \tag{9}$$

when this sum is well-defined and in $\mathbb{K}$, which is always the case when the semiring is *k-closed* or when $M$ is acyclic [11], the case of interest in what follows. There exists a general algorithm for computing the shortest-distance $d[q]$ in linear time $O(|Q| + (T_\oplus + T_\otimes)|E|)$, where $T_\oplus$ denotes the maximum time to compute $\oplus$ and $T_\otimes$ the time to compute $\otimes$ [11]. The algorithm is a generalization of Lawler's algorithm [8] to the case of an arbitrary semiring $\mathbb{K}$. It is based on a generalized relaxation of the outgoing transitions of each state of $M$ visited in reverse topological order [11].

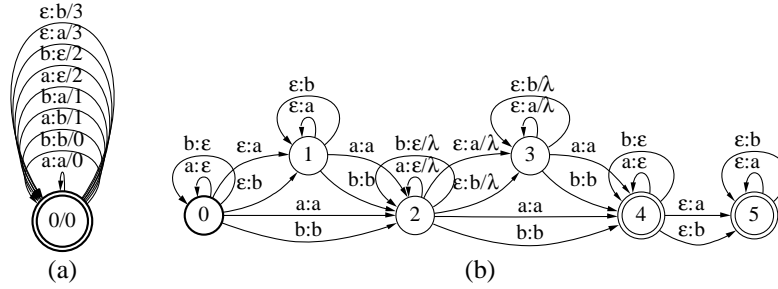

Figure 2: *Weighted transducers associated to two rational kernels. (a) Edit-distance kernel. (b) Gappy $N$-gram count kernel, with $N = 2$.*

### 3.4 Algorithm

Let $K$ be a rational kernel and let $T$ be the associated weighted transducer. Let $A$ and $B$ be two acyclic weighted automata. $A$ and $B$ may represent just two strings $x, y \in \Sigma^*$ or may be any other complex weighted acceptors. By definition of rational kernels (Eq.(5)) and the shortest-distance (Eq.(9)), $K(A, B)$ can be computed by:

1. Constructing the acyclic composed transducer $N = A \circ T \circ B$.
2. Computing $d[N]$, the shortest-distance from the initial states of $N$ to its final states using the shortest-distance algorithm described in the previous section.
3. Computing $\psi(d[N])$.

Thus, the total complexity of the algorithm is $O(|T||A||B| + \Phi)$, where $|T|$, $|A|$, and $|B|$ denote respectively the size of $T$, $A$ and $B$ and $\Phi$ the worst case complexity of computing $\psi(x)$, $x \in \mathbb{K}$. If we assume that $\Phi$ can be computed in constant time as in many applications, then the complexity of the computation of $K(A, B)$ is quadratic with respect to $A$ and $B$ is: $O(|T||A||B|)$.

### 3.5 Edit-distance kernels

Recently, several kernels, *string kernels*, have been introduced in computational biology for input vectors representing biological sequences [4, 19]. String kernels are specific instances of rational kernels. Fig.(2) (a) shows the weighted transducer over the tropical semiring associated to a classical type of string kernel. The kernel corresponds to an edit-distance based on a symbol substitution with cost 1, deletion with cost 2, and insertion of cost 3. All classical edit-distances can be represented by weighted transducers over the tropical semiring [13, 10]. The kernel computation algorithm just described can be used to compute efficiently the edit-distance of two strings or two sets of strings represented by automata. [3]

### 3.6 Rational kernels of the type $T \circ T^{-1}$

There exists a general method for constructing a positive definite and symmetric rational kernel from a weighted transducer $T$ when $\psi : \mathbb{K} \to \mathbb{R}$ is a semiring morphism – this implies in particular that $\mathbb{K}$ is commutative. Denote by $T^{-1}$ the *inverse* of $T$, that is the transducer obtained from $T$ by transposing the input and output labels of each transition. Then the composed transducer $S = T \circ T^{-1}$ is symmetric and, when it is regulated, defines

a positive definite symmetric rational kernel $K$. Indeed, since $\psi$ is a semiring morphism, by definition of composition:

$$K(x, y) = \psi([\![S]\!](x, y)) = \sum_z \psi([\![T]\!](x, z)) \cdot \psi([\![T]\!](y, z))$$

which shows that $K$ is symmetric. For any non-negative integer $n$ and for all $x, y$ we define a symmetric kernel $K_n$ by:

$$K_n(x, y) = \sum_{|z| \leq n} \psi([\![T]\!](x, z)) \cdot \psi([\![T]\!](y, z))$$

where the sum runs over all strings $z$ of length less or equal to $n$. Let $z_1, z_2, \ldots, z_m$ be an arbitrary ordering of these strings. For any $N \geq 1$ and any $x_1, \ldots, x_N \in \Sigma^*$, define the matrix $M$ by: $M_{ij} = K_n(x_i, x_j)$. Then, $M = AA^t$ with $A$ defined by $A_{ij} = \psi([\![T]\!](x_i, z_j))$. Thus, the eigenvalues of $M$ are all non-negative, which implies that $K_n$ is positive definite [1]. Since $K$ is a point-wise limit of $K_n$, $K(x, y) = \lim_{n \to \infty} K_n(x, y)$, $K$ is also definite positive [1].

## 4 Application to spoken-dialog classification

Rational kernels can be used in a variety of applications ranging from computational biology to optical character recognition. This section singles out one specific application, that of topic classification applied to the output of a speech recognizer. We will show how the use of weighted transducers rationalizes the design and optimization of kernels. Simple equations and graphs replace complex diagrams and intricate algorithms often used for the definition and analysis of string kernels.

As mentioned in the introduction, the output of a speech recognition system associated to a speech utterance is a weighted automaton called a *word lattice* representing a set of alternative sentences and their respective probabilities based on the models used. Rational kernels help address both the problem of handling variable-length sentences and that of applying a classification algorithm to such distributions of alternatives.

The traditional solution to sentence classification is the "bag-of-words" approach used in information retrieval. Because of the very large dimension of the input space, the use of large-margin classifiers such as SVMs [6] and AdaBoost [16] was found to be appropriate in such applications.

One approach adopted in various recent studies to measure the topic-similarity of two sentences consists of counting their common *non-contiguous $N$-grams*, i.e., their common substrings of $N$ words with possible insertions. These $N$-grams can be extracted explicitly from each sentence [16] or matched implicitly through a string kernel [9]. We will show that such kernels are rational and will describe how they can be easily constructed and computed using the general algorithms given in the previous section. More generally, we will show how rational kernels can be used to compute the expected counts of common non-contiguous $N$-grams of two weighted automata and thus define the topic-similarity of two lattices. This will demonstrate the simplicity, power, and flexibility of our framework for the design of kernels.

### 4.1 Application of $T \circ T^{-1}$ kernels

Consider a word lattice $L$ over the probability semiring. $L$ can be viewed as a probability distribution $P_L$ over all strings $s \in \Sigma^*$. The expected count or number of occurrences of an $N$-gram sequence $x$ in a string $s$ for the probability distribution $P_L$ is: $\sum_s P_L(s)|s|_x$, where $|s|_x$ denotes the number of occurrences of $x$ in $s$. It is easy to construct a weighted transducer $T_N$ that outputs the set of $N$-grams of an input lattice with their corresponding expected counts. Fig.(3) (a) shows that transducer, when the alphabet is reduced to $\Sigma = \{a, b\}$ and $N = 2$. Similarly, the transducer $T_{N,\lambda}$ of Fig.(3) (b) can be used to output non-contiguous or gappy $N$-grams with their expected counts. [4] Long gaps are penalized

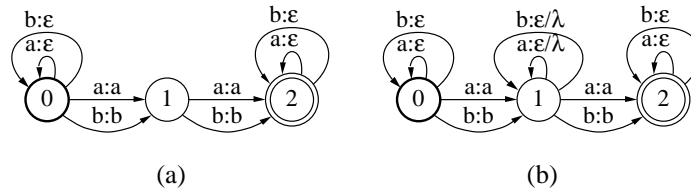

Figure 3: *N-gram transducers (N = 2) defined over the probability semiring. (a) Bigram counter transducer $T_2$. (b) Gappy bigram counter $T_{2,\lambda}$.*

with a decay factor $0 < \lambda \leq 1$: a gap of length $m$ reduces the count by $\lambda^m$. A transducer counting variable-length $N$-grams is obtained by simply taking the sum of these transducers: $T_{\leq N, \lambda} = \sum_{n \leq N} T_{n, \lambda}$.

In the remaining of this section, we will omit the subscript $N$ and $\lambda$ since our results are independent of the choice of these parameters. Thus the topic-similarity of two strings or lattices $X$ and $Y$ based on the expected counts of theirs common substrings is given by:

$$K(X, Y) = d[X \circ (T \circ T^{-1}) \circ Y] \qquad (10)$$

The kernel $K$ is of the type studied in section 3.6 and thus is symmetric and positive definite.

## 4.2 Computation

The specific form of the kernel $K$ and the associativity of composition provide us with several alternatives for computing $K$.

**General algorithm.** We can use the general algorithm described in Section 3.4 to compute $K$ by precomputing the transducer $T \circ T^{-1}$. Fig.(2)(b) shows the result of that composition in the case of gappy bigrams. Using that algorithm, the complexity of the computation of the kernel $K(X, Y)$ as described in the previous section is quadratic $O(|X||Y|)$. This particular example has been treated by *ad hoc* algorithms with a similar complexity, but that only work with strings [9, 5] and not with weighted automata or lattices.

**Other factoring.** Thanks to the associativity of composition, we can consider a different factoring of the composition cascade defining $K$:

$$K(X, Y) = d[(X \circ T) \circ (T^{-1} \circ Y)] \qquad (11)$$

This factoring suggests computing $X \circ T$ and $T^{-1} \circ Y$ first and then composing the resulting transducers rather than constructing $T \circ T^{-1}$. The choice between the two methods does not affect the overall time complexity of the algorithm, but in practice one method may be preferable over the other. We are showing elsewhere that in the specific case of the counting transducers such as those described in previous sections, the kernel computation can in fact be performed in linear time, that is in $O(|X| + |Y|)$, in particular by using the notion of *failure functions*.

## 4.3 Experimental results

We used the $T \circ T^{-1}$-type kernel with SVMs for call-classification in the spoken language understanding (SLU) component of the AT&T *How May I Help You* natural dialog system. In this system, users ask questions about their bill or calling plans and the objective is to assign a class to each question out of a finite set of 38 classes made of call-types and named entities such as *Billing Services*, or *Calling Plans*.

In our experiments, we used 7,449 utterances as our training data and 2,228 utterances as our test data. The feature space corresponding to our lattice kernel is that of all possible trigrams over a vocabulary of 5,405 words. Training required just a few minutes on a single processor of a 1GHz Intel Pentium processor Linux cluster with 2GB of memory and 256 KB cache. The implementation took only about a few hours and was entirely based on

the FSM library. Compared to the standard approach of using trigram counts over the best recognized sentence, our experiments with a trigram rational kernel showed a 15% reduction in error rate at a 30% rejection level.

## 5 Conclusion

In our classification experiments in spoken-dialog applications, we found rational kernels to be a very powerful exploration tool for constructing and generalizing highly efficient string and weighted automata kernels. In the design of learning machines such as SVMs, rational kernels give us access to the existing set of efficient and general weighted automata algorithms [13]. Prior knowledge about the task can be crafted into the kernel using graph editing tools or weighted regular expressions, in a way that is often more intuitive and easy to modify than complex matrices or formal algorithms.

## Footnotes

[1]We use a *matrix notation* for the definition of composition as opposed to a *functional notation*. This is a deliberate choice motivated by an improved readability in many applications.

[2]See [14, 12] for a detailed presentation of the algorithm including the use of a transducer filter for dealing with $\epsilon$-multiplicity in the case of non-idempotent semirings.

[3]We have proved and will present elsewhere a series of results related to kernels based on the notion of edit-distance. In particular, we have shown that the classical edit-distance $d$ with equal costs for insertion, deletion and substitution is not *negative definite* [1] and that the Gaussian kernel $\exp(-d)$ is not positive definite.

[4]The transducers shown in the figures of this section are all defined over the probability semiring, thus a transition corresponding to a gap in $T_{N,\lambda}$ is weighted by $\lambda$.

## References

[1] Christian Berg, Jens Peter Reus Christensen, and Paul Ressel. *Harmonic Analysis on Semigroups*. Springer-Verlag: Berlin-New York, 1984.

[2] Jean Berstel. *Transductions and Context-Free Languages*. Teubner Studienbucher: Stuttgart, 1979.

[3] Samuel Eilenberg. *Automata, Languages and Machines*, volume A-B. Academic Press, 1974.

[4] David Haussler. Convolution kernels on discrete structures. Technical Report UCSC-CRL-99-10, University of California at Santa Cruz, 1999.

[5] Ralf Herbrich. *Learning Kernel Classifiers*. MIT Press, Cambridge, 2002.

[6] Thorsten Joachims. Text categorization with support vector machines: learning with many relevant features. In *Proc. of ECML-98*. Springer Verlag, 1998.

[7] Werner Kuich and Arto Salomaa. *Semirings, Automata, Languages*. Number 5 in EATCS Monographs on Theoretical Computer Science. Springer-Verlag, Berlin, Germany, 1986.

[8] Eugene L. Lawler. *Combinatorial Optimization: Networks and Matroids*. Holt, Rinehart, and Winston, 1976.

[9] Huma Lodhi, John Shawe-Taylor, Nello Cristianini, and Christopher J. C. H. Watkins. Text classification using string kernels. In *NIPS*, pages 563–569, 2000.

[10] Mehryar Mohri. Edit-Distance of Weighted Automata. In Jean-Marc Champarnaud and Denis Maurel, editor, *Seventh International Conference, CIAA 2002*, volume to appear of *Lecture Notes in Computer Science*, Tours, France, July 2002. Springer-Verlag, Berlin-NY.

[11] Mehryar Mohri. Semiring Frameworks and Algorithms for Shortest-Distance Problems. *Journal of Automata, Languages and Combinatorics*, 7(3):321–350, 2002.

[12] Mehryar Mohri, Fernando C. N. Pereira, and Michael Riley. Weighted automata in text and speech processing. In *ECAI-96 Workshop, Budapest, Hungary*. ECAI, 1996.

[13] Mehryar Mohri, Fernando C. N. Pereira, and Michael Riley. The Design Principles of a Weighted Finite-State Transducer Library. *Theoretical Computer Science*, 231:17–32, January 2000. http://www.research.att.com/sw/tools/fsm.

[14] Fernando C. N. Pereira and Michael D. Riley. Speech recognition by composition of weighted finite automata. In Emmanuel Roche and Yves Schabes, editors, *Finite-State Language Processing*, pages 431–453. MIT Press, Cambridge, Massachusetts, 1997.

[15] Arto Salomaa and Matti Soittola. *Automata-Theoretic Aspects of Formal Power Series*. Springer-Verlag: New York, 1978.

[16] Robert E. Schapire and Yoram Singer. Boostexter: A boosting-based system for text categorization. *Machine Learning*, 39(2/3):135–168, 2000.

[17] Bernhard Scholkopf and Alex Smola. *Learning with Kernels*. MIT Press: Cambridge, MA, 2002.

[18] Vladimir N. Vapnik. *Statistical Learning Theory*. John Wiley & Sons, New-York, 1998.

[19] Chris Watkins. Dynamic alignment kernels. Technical Report CSD-TR-98-11, Royal Holloway, University of London, 1999.
